# Direct value-approximation for factored MDPs

**Dale Schuurmans and Relu Patrascu**
Department of Computer Science
University of Waterloo
{*dale,rpatrasc*}*@cs.uwaterloo.ca*

## Abstract

We present a simple approach for computing reasonable policies for factored Markov decision processes (MDPs), when the optimal value function can be approximated by a compact linear form. Our method is based on solving a single linear program that approximates the best linear fit to the optimal value function. By applying an efficient constraint generation procedure we obtain an iterative solution method that tackles concise linear programs. This direct linear programming approach experimentally yields a significant reduction in computation time over approximate value- and policy-iteration methods (sometimes reducing several hours to a few seconds). However, the quality of the solutions produced by linear programming is weaker—usually about twice the approximation error for the same approximating class. Nevertheless, the speed advantage allows one to use larger approximation classes to achieve similar error in reasonable time.

## 1 Introduction

Markov decision processes (MDPs) form a foundation for control in uncertain and stochastic environments and reinforcement learning. Standard methods such as value-iteration, policy-iteration and linear programming can be used to produce optimal control policies for MDPs that are expressed in explicit form; that is, the policy, value function and state transition model are all represented in a tabular manner that explicitly enumerates the state space. This renders the approaches impractical for all but toy problems. The real goal is to achieve solution methods that scale up reasonably in the size of the *state description*, not the size of the state space itself (which is usually either exponential or infinite).

There are two basic premises on which solution methods can scale up: (1) exploiting structure in the MDP model itself (i.e. structure in the reward function and the state transition model); and (2) exploiting structure in an approximate representation of the optimal value function (or policy). Most credible attempts at scaling-up have generally had to exploit both types of structure. Even then, it is surprisingly difficult to formulate an optimization method that can handle large state descriptions and yet simultaneously produce value functions or policies with small approximation errors, or errors that can be bounded tightly. In this paper we investigate a simple approach to determining approximately optimal policies based on a simple direct

linear programming approach. Specifically, the idea is to approximate the optimal value function by formulating a single linear program and exploiting structure in the MDP and the value function approximation to solve this linear program efficiently.

## 2   Preliminaries

We consider MDPs with finite state and action spaces and consider the goal of maximizing infinite horizon discounted reward. In this paper, states will be represented by vectors $\mathbf{x}$ of length $n$, where for simplicity we assume the state variables $x_1, ..., x_n$ are in $\{0, 1\}$; hence the total number of states is $N = 2^n$. We also assume there is a small finite set of actions $A = \{a_1, ..., a_\ell\}$. An MDP is defined by: (1) a state transition model $P(\mathbf{x}'|\mathbf{x}, a)$ which specifies the probability of the next state $\mathbf{x}'$ given the current state $\mathbf{x}$ and action $a$; (2) a reward function $R(\mathbf{x}, a)$ which specifies the immediate reward obtained by taking action $a$ in state $\mathbf{x}$; and (3) a discount factor $\gamma$, $0 \leq \gamma < 1$. The problem is to determine an optimal control policy $\pi^* : \mathbf{X} \to A$ that achieves maximum expected future discounted reward in every state.

To understand the standard solution methods it is useful to define some auxiliary concepts. For any policy $\pi$, the value function $V^\pi : \mathbf{X} \to I\!R$ denotes the expected future discounted reward achieved by policy $\pi$ in each state $\mathbf{x}$. It turns out that $V^\pi$ satisfies a fixed point relationship between the value of current states and the expected values of future states, given by a backup operator $V^\pi = B^\pi V^\pi$, where $B^\pi$ operates on arbitrary functions over the state space according to

$$(B^\pi f)(\mathbf{x}) = R(\mathbf{x}, \pi(\mathbf{x})) + \gamma \sum_{\mathbf{x}'} P(\mathbf{x}'|\mathbf{x}, \pi(\mathbf{x})) f(\mathbf{x}')$$

Another important backup operator is defined with respect to a fixed action $a$

$$(B^a f)(\mathbf{x}) = R(\mathbf{x}, a) + \gamma \sum_{\mathbf{x}'} P(\mathbf{x}'|\mathbf{x}, a) f(\mathbf{x}')$$

The action-value function $Q^\pi : \mathbf{X} \times A \to I\!R$ denotes the expected future discounted reward achieved by taking action $a$ in state $\mathbf{x}$ and following policy $\pi$ thereafter; which must satisfy $Q^\pi(\mathbf{x}, a) = B^a V^\pi$. Given an arbitrary function $f$ over states, the greedy policy $\pi_{gre}(f)$ with respect to $f$ is defined by

$$\pi_{gre}(f)(\mathbf{x}) = \arg\max_a (B^a f)(\mathbf{x})$$

Finally, if we let $\pi^*$ denote the optimal policy and $V^*$ denote its value function, we have the relationship $V^* = B^* V^*$, where $(B^* f)(\mathbf{x}) = \max_a (B^a f)(\mathbf{x})$. If, in addition, we define $Q^*(\mathbf{x}, a) = B^a V^*$ then we also have $\pi^*(\mathbf{x}) = \pi_{gre}(V^*)(\mathbf{x}) = \arg\max_a Q^*(\mathbf{x}, a)$. Given these definitions, the three fundamental methods for calculating $\pi^*$ can be formulated as:

**Policy iteration:** Start with an arbitrary policy $\pi^{(0)}$. Iterate $\pi^{(i+1)} \leftarrow \pi_{gre}(V^{\pi^{(i)}})$ until $\pi^{(i+1)} = \pi^{(i)}$. Return $\pi^* = \pi^{(i+1)}$.

**Value iteration:** Start with an arbitrary function $f^{(0)}$. Iterate $f^{(i+1)} \leftarrow B^* f^{(i)}$ until $\|f^{(i+1)} - f^{(i)}\|_\infty < tol$. Return $\pi^* = \pi_{gre}(f^{(i+1)})$.

**Linear programming:** Calculate $V^* = \arg\min_f \sum_{\mathbf{x}} f(\mathbf{x})$ subject to $f(\mathbf{x}) \geq (B^a f)(\mathbf{x})$ for all $a$ and $\mathbf{x}$. Return $\pi^* = \pi_{gre}(V^*)$.

All three methods can be shown to produce optimal policies for the given MDP [1, 10] even though they do so in very different ways. However, all three approaches share the same fundamental limitation that they do not scale up feasibly in $n$, the size of the state descriptions. Instead, all of these approaches work with explicit representations of the policies and value functions that are exponential in $n$.

# 3 Exploiting structure

To scale up to large state spaces it is necessary to exploit substantial structure in the MDP while also adopting some form of approximation for the optimal value function and policy. The two specific structural assumptions we consider in this paper are (1) factored MDPs and (2) linear value function approximations. Neither of these two assumptions alone is sufficient to permit efficient policy optimization for large MDPs. However, combined, the two assumptions allow approximate solutions to be obtained for problems involving trillions of states reasonably quickly.

## 3.1 Factored MDPs

In the spirit of [7, 8, 6] we define a *factored* MDP to be one that can be represented compactly by an additive reward function and a factored state transition model. Specifically, we assume the reward function decomposes as $R(\mathbf{x}, a) = \sum_{r=1}^{m} R_{a,r}(\mathbf{x}_{a,r})$ where each local reward function $R_{a,r}$ is defined on a small set of variables $\mathbf{x}_{a,r}$. We assume the state transition model $P(\mathbf{x}'|\mathbf{x}, a)$ can be represented by a set of dynamic Bayesian networks (DBNs) on state variables—one for each action—where each DBN defines a compact transition model on a directed bipartite graph connecting state variables in consecutive time steps. Let $\mathbf{x}_{a,i}$ denote the parents of successor variable $x_i'$ in the DBN for action $a$. To allow efficient optimization we assume the parent set $\mathbf{x}_{a,i}$ contains a small number of state variables from the previous time step. Given this model, the probability of a successor state $\mathbf{x}'$ given a predecessor state $\mathbf{x}$ and action $a$ is given by the product $P(\mathbf{x}'|\mathbf{x}, a) = \prod_{i=1}^{n} P(x_i'|\mathbf{x}_{a,i})$.

The main benefit of this factored representation is that it allows large MDPs to be encoded concisely: if the functions $R_{a,r}(\mathbf{x}_{a,r})$ and $P(x_i'|\mathbf{x}_{a,i})$ depend on a small number of variables, they can be represented by small tables and efficiently combined to determine $R(\mathbf{x}, a)$ and $P(\mathbf{x}'|\mathbf{x}, a)$. Unfortunately, as pointed out in [7], a factored MDP does not by itself yield a feasible method to determining optimal policies. The main problem is that, even if $P$ and $R$ are factored, the optimal value function generally does not have a compact representation (nor does the optimal policy). Therefore, obtaining an exact solution appears to require a return to explicit representations. However, it turns out that the factored MDP representation interacts very well with linear value function approximations.

## 3.2 Linear approximation

One of the central tenets to scaling up is to approximate the optimal value function rather than calculate it exactly. Numerous schemes have been investigated for approximating optimal value functions and policies in a compact representational framework, including: hierarchical decompositions [5], decision trees and diagrams [3, 12], generalized linear functions [1, 13, 4, 7, 8, 6], neural networks [2], and products of experts [11]. However, the simplest of these is generalized linear functions, which is the form we investigate below. In this case, we consider functions of the form $f(\mathbf{x}) = \sum_{j=1}^{k} w_j b_j(\mathbf{x}_j)$ where $b_1, ..., b_k$ are a fixed set of basis functions, and $\mathbf{x}_j$ denotes the variables on which basis $b_j$ depends. Combining linear functions with factored MDPs provides many opportunities for feasible approximation.

The first main benefit of combining linear approximation with factored MDPs is that the result of applying the backup operator $B^a$ to a linear function results in a compact representation for the action-value function. Specifically if we define

$g(\mathbf{x}, a) = (B^a f)(\mathbf{x})$ then we can rewrite it as

$$g(\mathbf{x}, a) = \sum_{r=1}^{m} R_{a,r}(\mathbf{x}_{a,r}) + \sum_{j=1}^{k} w_j c_{a,j}(\mathbf{x}_{a,\mathbf{j}})$$

where

$$c_{a,j}(\mathbf{x}_{a,\mathbf{j}}) = \gamma \sum_{\mathbf{x}'_j} P(\mathbf{x}'_j | a, \mathbf{x}_{a,\mathbf{j}}) b_j(\mathbf{x}'_j) \text{ and } \mathbf{x}_{a,\mathbf{j}} = \bigcup_{x'_i \in \mathbf{x}'_j} \mathbf{x}_{a,i}$$

That is, $\mathbf{x}_{a,i}$ are the parent variables of $x'_i$, and $\mathbf{x}_{a,\mathbf{j}}$ is the union of the parent variables of $x'_i \in \mathbf{x}'_j$. Thus, $c_{a,j}$ expresses the fact that in a factored MDP the expected future value of one component of the approximation depends only on the current state variables $\mathbf{x}_{a,\mathbf{j}}$ that are direct parents of the variables $\mathbf{x}'_j$ in $b_j$. If the MDP is sparsely connected then the variable sets in $g$ will not be much larger than those in $f$. The ability to represent the state-action value function in a compact linear form immediately provides a feasible implementation of the greedy policy for $f$, since $\pi_{gre}(f)(\mathbf{x}) = \arg\max_a g(\mathbf{x}, a)$ by definition of $\pi_{gre}$, and $g(\mathbf{x}, a)$ is efficiently determinable for each $\mathbf{x}$ and $a$. However, it turns out that this is not enough to permit feasible forms of approximate policy- and value-iteration to be easily implemented.

The main problem is that even though $B^a f$ has a factored form for fixed $a$, $B^* f$ does not and (therefore) neither does $\pi_{gre}(f)$. In fact, even if a policy $\pi$ were concisely represented, $B^\pi f$ would not necessarily have a compact form because $\pi$ usually depends on all the state variables and thus $P(\mathbf{x}' | \mathbf{x}, \pi(\mathbf{x})) = \prod_{i=1}^{n} P(x'_i | \mathbf{x}_{\pi(\mathbf{x}),i})$ becomes a product of terms that depend on all the state variables. Here [8, 6] introduce an additional assumption that there is a special "default" action $a_d$ for the MDP such that all other actions $a$ have a factored transition model $P(\cdot | \cdot, a)$ that differs from $P(\cdot | \cdot, a_d)$ only on a small number of state variables. This allows the greedy policy $\pi_{gre}(f)$ to have a compact form and moreover allows $B^{\pi_{gre}(f)} f$ to be concisely represented. With some effort, it then becomes possible to formulate feasible versions of approximate policy- and value-iteration [8, 6].

**Approximate policy iteration:** Start with default policy $\pi^{(0)}(\mathbf{x}) = a_d$. Iterate $f^{(i)} \leftarrow \arg\min_f \max_{\mathbf{x}} |f(\mathbf{x}) - (B^{\pi^{(i)}} f)(\mathbf{x})|$ , $\pi^{(i+1)} \leftarrow \pi_{gre}(f^{(i)})$ until $\pi^{(i+1)} = \pi^{(i)}$.

**Approximate value iteration:** Start with arbitrary $f^{(0)}$. Iterate $\pi^{(i)} \leftarrow \pi_{gre}(f^{(i)})$ , $f^{(i+1)} \leftarrow \arg\min_f \max_{\mathbf{x}} |f(\mathbf{x}) - (B^{\pi^{(i)}} f)(\mathbf{x})|$ until $\|f^{(i+1)} - f^{(i)}\|_\infty < tol$.

The most expensive part of these iterative algorithms is determining $\arg\min_f \max_{\mathbf{x}} |f(\mathbf{x}) - (B^{\pi^{(i)}} f)(\mathbf{x})|$ which involves solving a linear program $\min_{\mathbf{w}, \epsilon} \epsilon$ subject to $-\epsilon \leq f_{\mathbf{w}}(\mathbf{x}) - (B^\pi f_{\mathbf{w}})(\mathbf{x}) \leq \epsilon$ for all $\mathbf{x}$. This linear program is problematic because it involves an exponential number of constraints. A central achievement of [6] is to show that this system of constraints can be encoded by an equivalent system of constraints that has a much more compact form. The idea behind this construction is to realize that searching for the max or a min of a linear function with a compact basis can be conducted in an organized fashion, and such an organized search can be encoded in an equally concise constraint system. This construction allows approximate solutions to MDPs with up to $n = 40$ state variables (1 trillion states) to be generated in under 7.5 hours using approximate policy iteration [6].[1]

Our main observation is that if one has to solve linear programs to conduct the approximate iterations anyway, then it might be much simpler and more efficient to approximate the linear programming approach directly.

## 4 Approximate linear programming

Our first idea is simply to observe that a factored MDP and linear value approximation immediately allow one to directly solve the linear programming approximation to the optimal value function, which is given by

$$\min_f \sum_{\mathbf{x}} f(\mathbf{x}) \text{ subject to } f(\mathbf{x}) - (B^a f)(\mathbf{x}) \geq 0 \text{ for all } \mathbf{x} \text{ and } a$$

where $f$ is restricted to a linear form over a fixed basis. In fact, it is well known [1, 2] that this yields a linear program in the basis weights $\mathbf{w}$. However, what had not been previously shown is that given a factored MDP, an equivalent linear program of feasible size could be formulated. Given the results of [6] outlined above this is now easy to do. First, one can show that the minimization objective can be encoded compactly

$$\sum_{\mathbf{x}} f(\mathbf{x}) = \sum_{\mathbf{x}} \sum_{j=1}^{k} w_j b_j(\mathbf{x}_j)$$

$$= \sum_{j=1}^{k} w_j y_j \quad \text{where } y_j = 2^{n-|\mathbf{x}_j|} \sum_{\mathbf{x}_j} b_j(\mathbf{x}_j)$$

Here the $y_j$ components can be easily precomputed by enumerating assignments to the small sets of variables in basis functions. Second, as we have seen, the exponentially many constraints have a structured form. Specifically $f(\mathbf{x}) - (B^a f)(\mathbf{x})$ can be represented as

$$f(\mathbf{x}) - (B^a f)(\mathbf{x}) = \sum_{j=1}^{k} w_j \left( b_j(\mathbf{x}_j) - c_{a,j}(\mathbf{x}_{a,\mathbf{j}}) \right) - \sum_r R_{a,r}(\mathbf{x}_{a,r})$$

which has a simple basis representation that allows the technique of [6] to be used to encode a constraint system that enforces $f(\mathbf{x}) - (B^a f)(\mathbf{x}) \geq 0$ for all $x$ and $a$ without enumerating the state space for each action.

We implemented this approach and tested it on some of the test problems from [6]. In these problems there is a directed network of computer systems $x_1, ..., x_n$ where each system is either up ($x_i = 1$) or down ($x_i = 0$). Systems can spontaneously go down with some probability at each step, but this probability is increased if an immediately preceding machine in the network is down. There are $n + 1$ actions: do nothing (the default) and reboot machine $i$. The reward in a state is simply the sum of systems that are up, with a bonus reward of 1 if system 1 (the server) is up. I.e., $R(\mathbf{x}) = 2x_1 + \sum_{i=2}^{n} x_i$. We considered the network architectures shown in Figure 1 and used the transition probabilities $P(x_i' = 1|x_i, parent(x_i), a = i) = 0.95$ and $P(x_i' = 1|x_i, parent(x_i), a \neq i) = 0.9$ if $x_i = parent(x_i) = 1$; 0.67 if $x_i = 1$ and $parent(x_i) = 0$; and 0.01 if $x_i = 0$. The discount factor was $\gamma = 0.95$. The first basis functions we considered were just the indicators on each variable $x_i$ plus a constant basis function (as reported in [6]).

The results for two network architectures are shown in Figure 1. Our approximate linear programming method is labeled ALP and is compared to the approximate

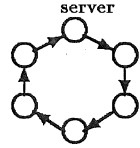

server

| | | $n =$ | 12 | 16 | 20 | 24 | 28 | 32 | 36 | 40 |
|---|---|---|---|---|---|---|---|---|---|---|
| | | $N =$ | 4e3 | 6e4 | 1e6 | 2e7 | 3e8 | 4e9 | 7e10 | 1e12 |
| | time | API[6][2] | 7m | 30m | 50m | 1.3h | 1.9h | 3h | 4.5h | 7.5h |
| | | APIgen | 39s | 1.5m | 2.3m | 4.0m | 6.5m | 13m | 22m | 28m |
| | | ALP | 4.5s | 23s | 1.4m | 4.1m | 10m | 23m | 47m | 2.4h |
| | | ALPgen | 0.7s | 1.2s | 1.8s | 2.6s | 3.5s | 4.5s | 5.9s | 7.0s |
| | | ALPgen2 | 14s | 37s | 1.2m | 2.8m | 4.7m | 6.4m | 12m | 17m |
| | constraints | APIgen | 420 | 777 | 921 | 1270 | 1591 | 2747 | 4325 | 4438 |
| | | ALP | 1131 | 2023 | 3171 | 4575 | 6235 | 8151 | 10K | 13K |
| | | ALPgen | 38 | 50 | 62 | 74 | 86 | 98 | 110 | 122 |
| | | ALPgen2 | 166 | 321 | 514 | 914 | 1223 | 1433 | 1951 | 2310 |
| | UB Bellman / Rmax | API[6][2] | 0.30 | 0.33 | 0.34 | 0.35 | 0.36 | 0.36 | 0.37 | 0.38 |
| | | APIgen | 0.36 | 0.34 | 0.33 | 0.33 | 0.32 | 0.32 | 0.32 | 0.31 |
| | | ALP(gen) | 0.85 | 0.82 | 0.80 | 0.78 | 0.78 | 0.77 | 0.76 | 0.76 |
| | | ALPgen2 | 0.12 | 0.14 | 0.08 | 0.08 | 0.10 | 0.08 | 0.07 | 0.07 |

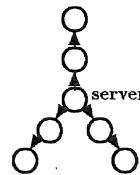

server

| | | $n =$ | 13 | 16 | 22 | 28 | 34 | 40 |
|---|---|---|---|---|---|---|---|---|
| | | $N =$ | 8e4 | 6e4 | 4e6 | 3e8 | 2e10 | 1e12 |
| | time | API[6][2] | 5m | 15m | 50m | 1.3h | 2.7h | 5h |
| | | APIgen | 28s | 1.6m | 3.9m | 12m | 23m | 33m |
| | | ALP | 0.7s | 1.6s | 6.0s | 20s | 56s | 2.2m |
| | | ALPgen | 0.7s | 1.0s | 1.5s | 2.4s | 3.4s | 4.7s |
| | | ALPgen2 | 17s | 33s | 1.9m | 5.4m | 9.6m | 23m |
| | constraints | APIgen | 363 | 952 | 1699 | 3792 | 6196 | 7636 |
| | | ALP | 729 | 1089 | 2025 | 3249 | 4761 | 6561 |
| | | ALPgen | 50 | 69 | 90 | 114 | 135 | 162 |
| | | ALPgen2 | 261 | 381 | 826 | 1505 | 1925 | 3034 |
| | UB Bellman / Rmax | API[6][2] | 0.27 | 0.29 | 0.32 | 0.34 | 0.35 | 0.36 |
| | | APIgen | 0.50 | 0.46 | 0.42 | 0.39 | 0.38 | 0.37 |
| | | ALP(gen) | 0.96 | 0.82 | 0.78 | 0.78 | 0.77 | 0.76 |
| | | ALPgen2 | 0.21 | 0.22 | 0.15 | 0.06 | 0.07 | 0.03 |

Figure 1: Experimental results (timings on a 750MHz PIII processor, except [2])

policy iteration strategy API described in [6]. Since we did not have the specific probabilities used in [6] and could only estimate the numbers for API from graphs presented in the paper, this comparison is only meant to be loosely indicative of the general run times of the two methods on such problems. (Perturbing the probability values did not significantly affect our results, but we implemented APIgen for comparison.) As in [6] our implementation is based on Matlab, using CPLEX to solve linear programs. Our preliminary results appear to support the hypothesis that direct linear programming can be more efficient than approximate policy iteration on problems of this type. A further advantage of the linear programming approach is that it is simpler to program and involves solving only one LP. More importantly, the direct LP approach does not require the MDP to have a special default action since the action-value function can be directly extracted using $\pi_{gre}(f)(\mathbf{x}) = \arg\max_a g(\mathbf{x}, a)$ and $g$ is easily recoverable from $f$.

Before discussing drawbacks, we note that it is possible to solve the linear program even more efficiently by iteratively generating constraints as needed. This is now possible because factored MDPs and linear value approximations allow an efficient search for the maximally violated constraints in the linear program, which provides an effective way of generating concise linear programs that can be solved much more efficiently than those formulated above. Specifically, the procedure ALPgen exploits the feasible search techniques for minimizing linear functions discussed previously to efficiently generate a small set of critical constraints, which is iteratively grown until the final solution is identified; see Figure 2.

**ALPgen**
    Start with $f^{(0)} = 0$ and *constraints* $= \emptyset$
    Loop
        For each $a \in A$, compute $\mathbf{x}^a \leftarrow \arg\min_{\mathbf{x}} f^{(i)}(\mathbf{x}) - (B^a f^{(i)})(\mathbf{x})$
        *constraints* $\leftarrow$ *constraints* $\bigcup \{constraint(\mathbf{x}^{a_1}), ..., constraint(\mathbf{x}^{a_k})\}$
        Solve $f^{(i+1)} \leftarrow \min_f \sum_{\mathbf{x}} f(\mathbf{x})$ subject to *constraints*
    Until $\min_x f^{(i)}(\mathbf{x}) - (B^a f^{(i)})(\mathbf{x}) \geq 0 - tol$ for all $a$
    Return $g(\cdot, a) = B^a f$ for each $a$, to represent the greedy policy

Figure 2: ALPgen procedure

The rationale for this procedure is that the main bottleneck in the previous methods is generating the constraints, not solving the linear programs [6]. Since only a small number of constraints are active at a solution and these are likely to be the most violated near the solution, adding only most violated constraints appears to be a useful way to proceed. Indeed, Figure 1 shows that ALPgen produces the same approximate solutions as ALP in a tiny fraction of the time. In the most extreme case ALPgen produces an approximate solution in 7 seconds while other methods take several hours on the same problem. The reason for this speedup is explained by the results which show the numbers of constraints generated by each method. Further investigation is also required to fully outline the robustness of the constraint generation method. In fact, one cannot guarantee that a greedy constraint generation scheme like the one proposed here will always produce a feasible number of constraints [9]. Nevertheless, the potential benefits of conservatively generating constraints as needed seem to be clear. Of course, the main drawback of the direct linear programming approach over approximate policy iteration is that ALP incurs larger approximation errors than API.

## 5 Bounding approximation error

It turns out that neither API nor ALP are guaranteed to return the best linear approximation to the true value function. Nevertheless, it is possible to efficiently calculate bounds on the approximation errors of these methods, again by exploiting the structure of the problem: A well known result [14] asserts that $\max_{\mathbf{x}} V^*(\mathbf{x}) - V^{\pi_{gre}(f)}(\mathbf{x}) \leq \frac{\gamma}{1-\gamma} \max_x f(\mathbf{x}) - (B^* f)(\mathbf{x})$ (where in our case $f \geq V^*$). This upper bound can in turn be bounded by a quantity that is feasible to calculate: $\max_x f(\mathbf{x}) - (B^* f)(\mathbf{x}) = \max_{\mathbf{x}} \min_a f(\mathbf{x}) - (B^a f)(\mathbf{x}) \leq \min_a \max_{\mathbf{x}} f(\mathbf{x}) - (B^a f)(\mathbf{x})$. Thus an upper bound on the error from the optimal value function can be calculated by performing an efficient search for $\max_{\mathbf{x}} f(\mathbf{x}) - (B^a f)(\mathbf{x})$ for each $a$.

Figure 1 shows that the measurable error quantity, $\max_{\mathbf{x}} f(\mathbf{x}) - (B^a f)(\mathbf{x})$ (reported as UB Bellman) is about a factor of two larger for the linear programming approach than for approximate policy iteration on the same basis. In this respect, API appears to have an inherent advantage (although in the limit of an exhaustive basis both approaches converge to the same optimal value). To get an indication of the computational cost required for ALPgen to achieve a similar bound on approximation error, we repeated the same experiments with a larger basis set that included all four indicators between pairs of connected variables. The results for this model are reported as ALPgen2, and Figure 1 shows that, indeed, the bound on approximation error is reduced substantially—but at the predictable cost of a sizable increase in computation time. However, the run times are still appreciably smaller than the policy iteration methods.

Paradoxically, linear programming seems to offer computational advantages over policy and value iteration in the context of approximation, even though it is widely held to be an inferior solution strategy for explicitly represented MDPs.

## Footnotes

[1] It turns out that approximate value iteration is less effective because it takes more iterations to converge, and in fact can diverge in theory [6, 13].

[2] These numbers are estimated from graphs in [6]. The exact probabilities and computer used for the simulations were not reported in that paper, so we cannot assert an exact comparison. However, perturbed probabilities have little effect on the performance of the methods we tried, and it seems that overall this is a loosely representative comparison of the general performance of the various algorithms on these problems.

# References

[1] D. Bertsekas. *Dynamic Programming and Optimal Control*, volume 2. Athena Scientific, 1995.

[2] D. Bertsekas and J. Tsitsiklis. *Neuro-Dynamic Programming*. Athena Scientific, 1996.

[3] C. Boutilier, R. Dearden, and M. Goldszmidt. Stochastic dynamic programming with factored representations. *Artificial Intelligence*, 2000.

[4] J. Boyan. Least-squares temporal difference learning. In *Proceedings ICML*, 1999.

[5] T. Dietterich. Hierarchical reinforcement learning with the MAXQ value function decomposition. *JAIR*, 13:227–303, 2000.

[6] C. Guestrin, D. Koller, and R. Parr. Max-norm projection for factored MDPs. In *Proceedings IJCAI*, 2001.

[7] D. Koller and R. Parr. Computing factored value functions for policies in structured MDPs. In *Proceedings IJCAI*, 1999.

[8] D. Koller and R. Parr. Policy iteration for factored MDPs. In *Proceedings UAI*, 2000.

[9] R. Martin. *Large Scale Linear and Integer Optimization*. Kluwer, 1999.

[10] M. Puterman. *Markov Decision Processes: Discrete Dynamic Programming*. Wiley, 1994.

[11] B. Sallans and G. Hinton. Using free energies to represent Q-values in a multiagent reinforcement learning task. In *Proceedings NIPS*, 2000.

[12] R. St-Aubin, J. Hoey, and C. Boutilier. APRICODD: Approximating policy construction using decision diagrams. In *Proceedings NIPS*, 2000.

[13] B. Van Roy. *Learning and value function approximation in complex decision processes*. PhD thesis, MIT, EECS, 1998.

[14] R. Williams and L. Baird. Tight performance bounds on greedy policies based on imperfect value functions. Technical report, Northeastern University, 1993.
